# FloatBoost Learning for Classification

**Stan Z. Li**[*]
Microsoft Research Asia
Beijing, China

**ZhenQiu Zhang**[†]
Institute of Automation
CAS, Beijing, China

**Heung-Yeung Shum**
Microsoft Research Asia
Beijing, China

**HongJiang Zhang**
Microsoft Research Asia
Beijing, China

## Abstract

AdaBoost [3] minimizes an upper error bound which is an exponential function of the margin on the training set [14]. However, the ultimate goal in applications of pattern classification is always minimum error rate. On the other hand, AdaBoost needs an effective procedure for learning weak classifiers, which by itself is difficult especially for high dimensional data. In this paper, we present a novel procedure, called FloatBoost, for learning a better boosted classifier. FloatBoost uses a backtrack mechanism after each iteration of AdaBoost to remove weak classifiers which cause higher error rates. The resulting float-boosted classifier consists of fewer weak classifiers yet achieves lower error rates than AdaBoost in both training and test. We also propose a statistical model for learning weak classifiers, based on a stagewise approximation of the posterior using an overcomplete set of scalar features. Experimental comparisons of FloatBoost and AdaBoost are provided through a difficult classification problem, face detection, where the goal is to learn from training examples a highly nonlinear classifier to differentiate between face and nonface patterns in a high dimensional space. The results clearly demonstrate the promises made by FloatBoost over AdaBoost.

## 1 Introduction

Nonlinear classification of high dimensional data is a challenging problem. While designing such a classifier is difficult, AdaBoost learning methods, introduced by Freund and Schapire [3], provides an effective stagewise approach: It learns a sequence of more easily learnable "weak classifiers", and boosts them into a single strong classifier by a linear combination of them. It is shown that the AdaBoost learning minimizes an upper error bound which is an exponential function of the margin on the training set [14].

Boosting learning originated from the PAC (probably approximately correct) learning theory [17, 6]. Given that weak classifiers can perform slightly better than random guessing

---

[*]http://research.microsoft.com/~szli
[†]The work presented in this paper was carried out at Microsoft Research Asia.

on every distribution over the training set, AdaBoost can provably achieve arbitrarily good bounds on its training and generalization errors [3, 15]. It is shown that such simple weak classifiers, when boosted, can capture complex decision boundaries [1].

Relationships of AdaBoost [3, 15] to functional optimization and statistical estimation are established recently. A number of gradient boosting algorithms are proposed [4, 8, 21]. A significant advance is made by Friedman *et al.* [5] who show that the AdaBoost algorithms minimize an exponential loss function which is closely related to Bernoulli likelihood.

In this paper, we address the following problems associated with AdaBoost:

1. AdaBoost minimizes an exponential (some another form of ) function of the margin over the training set. This is for convenience of theoretical and numerical analysis. However, the ultimate goal in applications is always minimum error rate. A strong classifier learned by AdaBoost may not necessarily be best in this criterion. This problem has been noted, eg by [2], but no solutions have been found in literature.

2. An effective and tractable algorithm for learning weak classifiers is needed. Learning the optimal weak classifier, such as the log posterior ratio given in [15, 5], requires estimation of densities in the input data space. When the dimensionality is high, this is a difficult problem by itself.

We propose a method, called FloatBoost (Section 3), to overcome the first problem. Float-Boost incorporates into AdaBoost the idea of Floating Search originally proposed in [11] for feature selection. A backtrack mechanism therein allows deletion of those weak classifiers that are non-effective or unfavorable in terms of the error rate. This leads to a strong classifier consisting of fewer weak classifiers. Because deletions in backtrack is performed according to the error rate, an improvement in classification error is also obtained. To solve the second problem above, we provide a statistical model (Section 4) for learning weak classifiers and effective feature selection in high dimensional feature space. A base set of weak classifiers, defined as the log posterior ratio, are derived based on an overcomplete set of scalar features. Experimental results are presented in (Section 5) using a difficult classification problem, face detection. Comparisons are made between FloatBoost and Ad-aBoost in terms of the error rate and complexity of boosted classifier. Results clear show that FloatBoost yields a strong classifier consisting of fewer weak classifiers yet achieves lower error rates.

## 2   AdaBoost Learning

In this section, we give a brief description of AdaBoost algorithm, in the notion of Real-Boost [15, 5], as opposed to the original discrete AdaBoost [3].

For two class problems, a set of $N$ labelled training examples is given as $(x_1, y_1), \ldots, (x_N, y_N)$, where $y_i \in \{+1, -1\}$ is the class label associated with example $x_i \in \mathbb{R}^n$. A stronger classifier is a linear combination of $M$ weak classifiers

$$H_M(x) = \sum_{m=1}^{M} h_m(x) \tag{1}$$

In this real version of AdaBoost, the weak classifiers can take a real value, $h_m(x) \in \mathbb{R}$, and have absorbed the coefficients needed in the discrete version (there, $h_m(x) \in -1, +1$). The class label for $x$ is obtained as $H(x) = \text{sign}[H_M(x)]$ while the magnitude $|H_M(x)|$ indicates the confidence. Every training example is associated with a weight. During the learning process, the weights are updated dynamically in such a way that more emphasis is

placed on hard examples which are erroneously classified previously. It is important for the original AdaBoost. However, recent studies [4, 8, 21] show that the artificial operation of explicit re-weighting is unnecessary and can be incorporated into a functional optimization procedure of boosting.

---

0. (Input)
    (1) Training examples $\mathcal{Z} = \{(x_1, y_1), \ldots, (x_N, y_N)\}$,
      where $N = a + b$; of which $a$ examples have $y_i = +1$
      and $b$ examples have $y_i = -1$;
    (2) The maximum number $M_{\max}$ of weak classifiers to be combined;
1. (Initialization)
    $w_i^{(0)} = \frac{1}{2a}$ for those examples with $y_i = +1$ or
    $w_i^{(0)} = \frac{1}{2b}$ for those examples with $y_i = -1$.
    $M = 0$;
2. (Forward Inclusion)
  while $M < M_{\max}$
    (1) $M \leftarrow M + 1$;
    (2) Choose $h_M$ according to Eq.4;
    (3) Update $w_i^{(M)} \leftarrow \exp[-y_i H_M(x_i)]$, and normalize to $\sum_i w_i^{(M)} = 1$;
3. (Output)
    $H(x) = \text{sign}[\sum_{m=1}^{M} h_m(x)]$.

---

Figure 1: RealBoost Algorithm.

An error occurs when $H(x) \neq y$, or $y H_M(x) < 0$. The "margin" of an example $(x, y)$ achieved by $h(x) \in \mathbb{R}$ on the training set examples is defined as $yh(x)$. This can be considered as a measure of the confidence of the $h$'s prediction. The upper bound on classification error achieved by $H_M$ can be derived as the following exponential loss function [14]

$$J(H_M) = \sum_i e^{-y_i H_M(x_i)} \tag{2}$$

AdaBoost construct $h(x)$ by stage-wise minimization of Eq.(2). Given the current $H_{M-1}(x) = \sum_{m=1}^{M-1} h_m(x)$, the best $h_M(x)$ for the new strong classifier $H_M(x) = H_{M-1}(x) + h_M(x)$ is the one which leads to the minimum cost

$$h_M = \arg\min_{h^\dagger} J(H_{M-1}(x) + h^\dagger(x)) \tag{3}$$

It is shown in [15, 5] that the minimizer is

$$h_M(x) = \frac{1}{2} \log \frac{P(y = +1 \mid x, w^{(M-1)})}{P(y = -1 \mid x, w^{(M-1)})} \tag{4}$$

where $w^{(M-1)}$ are the weights given at time $M$. Using $P(y \mid x, w) = P(x \mid y, w)P(y)$ and letting

$$L_M(x) = \frac{1}{2} \log \frac{p(x \mid y = +1, w)}{p(x \mid y = -1, w)} \tag{5}$$

$$T = \frac{1}{2} \left[ \log \frac{P(y = +1)}{P(y = -1)} \right] \tag{6}$$

we arrive

$$h_M(x) = L_M(x) + T \tag{7}$$

The half log likelihood ratio $L(x)$ is learned from the training examples of the two classes, and the threshold $T$ is determined by the log ratio of prior probabilities. $T$ can be adjusted

to balance between detection rate and false alarm (ROC curve). The algorithm is shown in Fig.1 (Note: Re-weight formula in this description is equivalent to the multiplicative rule in the original form of AdaBoost [3, 15]). In Section 4, we will present an model for approximating $P(x \mid y, w^{(M-1)})$.

## 3  FloatBoost Learning

FloatBoost backtracks after the newest weak classifier $h_M$ is added and delete unfavorable weak classifiers $h_m$ from the ensemble (1), following the idea of Floating Search [11]. Floating Search [11] is originally aimed to deal with non-monotonicity of straight sequential feature selection, non-monotonicity meaning that adding an additional feature may lead to drop in performance. When a new feature is added, backtracks are performed to delete those features that cause performance drops. Limitations of sequential feature selection is thus amended, improvement gained with the cost of increased computation due to the extended search.

---

0. (Input)
    (1) Training examples $\mathcal{Z} = \{(x_1, y_1), \ldots, (x_N, y_N)\}$,
       where $N = a + b$; of which $a$ examples have
       $y_i = +1$ and $b$ examples have $y_i = -1$;
    (2) The maximum number $M_{\max}$ of weak classifiers;
    (3) The error rate $\varepsilon(H_M)$, and the acceptance threshold $\varepsilon^*$.
1. (Initialization)
    (1) $w_i^{(0)} = \frac{1}{2a}$ for those examples with $y_i = +1$ or
       $w_i^{(0)} = \frac{1}{2b}$ for those examples with $y_i = -1$;
    (2) $\varepsilon_m^{\min}$ =max-value (for $m = 1, \ldots, M_{\max}$),
       $M = 0, \mathcal{H}_0 = \{\}$.
2. (Forward Inclusion)
    (1) $M \leftarrow M + 1$;
    (2) Choose $h_M$ according to Eq.4;
    (3) Update $w_i^{(M)} \leftarrow \exp[-y_i H_M(x_i)]$, and normalize to $\sum_i w_i^{(M)} = 1$;
    (4) $\mathcal{H}_M = \mathcal{H}_{M-1} \cup \{h_M\}$; If $\varepsilon_M^{\min} > \varepsilon(H_M)$, then $\varepsilon_M^{\min} = \varepsilon(H_M)$;
3. (Conditional Exclusion)
    (1) $h' = \arg\min_{h \in \mathcal{H}_M} \varepsilon(H_M - h)$;
    (2) If $\varepsilon(H_M - h') < \varepsilon_{M-1}^{\min}$, then
       (a) $\mathcal{H}_{M-1} = \mathcal{H}_M - h'$;
         $\varepsilon_{M-1}^{\min} = \varepsilon(H_M - h'); M = M - 1$;
       (b) $H_M = \sum_{h \in \mathcal{H}_M} h$;
       (c) goto 3.(1);
    (3) else
       (a) if $M = M_{\max}$ or $J(\mathcal{H}_M) < J^*$, then goto 4;
       (b) $w_i^{(M)} \leftarrow \exp[-y_i H_M(x_i)]$; goto 2.(1);
4. (Output)
    $H(x) = \text{sign}[\sum_{h(x) \in \mathcal{H}_M} h(x)]$.

---

Figure 2: FloatBoost Algorithm.

The FloatBoost procedure is shown in Fig.2 Let $\mathcal{H}_M = \{h_1, \ldots, h_M\}$ be the so-far-best set of $M$ weak classifiers; $\varepsilon(H_M)$ be the error rate achieved by $H_M(x) = \sum_{m=1}^{M} h_m(x)$ (or a weighted sum of missing rate and false alarm rate which is usually the criterion in one-class detection problem); $\varepsilon_m^{\min}$ be the minimum error rate achieved so far with an ensemble of $m$ weak classifiers.

In Step 2 (forward inclusion), given already selected, the best weak classifier is added one

at a time, which is the same as in AdaBoost. In Step 3 (conditional exclusion), FloatBoost removes the least significant weak classifier from $\mathcal{H}_M$, subject to the condition that the removal leads to a lower error rate $\varepsilon_{M-1}^{\min}$. These are repeated until no more removals can be done. The procedure terminates when the risk on the training set is below $J^*$ or the maximum number $M_{\max}$ is reached.

Incorporating the conditional exclusion, FloatBoost renders both effective feature selection and classifier learning. It usually needs fewer weak classifiers than AdaBoost to achieve the same error rate $\varepsilon$.

## 4  Learning Weak Classifiers

The section presents a method for computing the log likelihood in Eq.(5) required in learning optimal weak classifiers. Since deriving a weak classifier in high dimensional space is a non-trivial task, here we provide a statistical model for stagewise learning of weak classifiers based on some scalar features. A scaler feature $k$ of $x$ is computed by a transform from the $n$-dimensional data space to the real line, $z_k(x) \in \mathbb{R}$. A feature can be the coefficient of, say, a wavelet transform in signal and image processing. If projection pursuit is used as the transform, $z_k(x)$ is simply the $k$-th coordinate of $x$. A dictionary of $K$ candidate scalar features can be created $\mathcal{Z} = \{z_1(x), \ldots, z_K(x)\}$. In the following, we use $z_{(m)}$ to denote the feature selected in the $m$-th stage, while $z_k(x)$ is the feature computed from $x$ using the $k$-th transform.

Assuming that $\mathcal{Z}$ is an over-complete basis, a set of candidate weak classifiers for the optimal weak classifier (7) can be designed in the following way: First, at stage $M$ where $M-1$ features $z_{(1)}, z_{(2)}, \ldots, z_{(M-1)}$ have been selected and the weight is given as $w^{(M-1)}$, we can approximate $p(x \mid y, w^{(M-1)})$ by using the distributions of $M$ features

$$
\begin{aligned}
p(x \mid y, w^{(M-1)}) &\approx p(z_{(1)}, z_{(2)}, \ldots, z_{(M-1)}, z_k \mid y, w^{(M-1)}) \quad (8) \\
&= p(z_{(1)} \mid y, w^{(M-1)})\, p(z_{(2)} \mid y, z_{(1)}, w^{(M-1)}) \cdots \\
&\quad p(z_{(M-1)} \mid y, z_{(1)}, \ldots, z_{(M-2)}, w^{(M-1)}) \\
&\quad p(z_k \mid y, z_{(1)}, \ldots, z_{(M-1)}, w^{(M-1)}) \quad (9)
\end{aligned}
$$

Because $\mathcal{Z}$ is an over-complete basis set, the approximation is good enough for large enough $M$ and when the $M$ features are chosen appropriately.

Note that $p(z_{(m)} \mid y, z_{(1)}, \ldots, z_{(m-1)})$ is actually $p(z_{(m)} \mid y, w^{(m-1)})$ because $w^{(m)}$ contains the information about entire history of $w$ and accounts for the dependencies on $z_{(1)}, \ldots, z_{(m-1)}$. Therefore, we have

$$
\begin{aligned}
p(x \mid y, w^{(M-1)}) &\approx p(z_{(1)} \mid y, w^{(0)})\, p(z_{(2)} \mid y, w^{(1)}) \cdots \quad (10) \\
&\quad p(z_{(M-1)} \mid y, w^{(M-2)}) p(z_k \mid y, w^{(M-1)}) \quad (11)
\end{aligned}
$$

On the right-hand side of the above equation, all conditional densities are fixed except the last one $p(z_k \mid y, w^{(M-1)})$. Learning the best weak classifier at stage $M$ is to choose the best feature $z_{(M)}$ for $z_k$ such that $J$ is minimized according to Eq.(3).

The conditional probability densities $p(z_k \mid y, w^{(M-1)})$ for the positive class $y = +1$ and the negative class $y = -1$ can be estimated using the histograms computed from the weighted voting of the training examples using the weights $w^{(M-1)}$. Let

$$
L_k^{(M)}(x) = \frac{1}{2} \log \frac{p(z_k \mid y = +1, w^{(M-1)})}{p(z_k \mid y = -1, w^{(M-1)})} \quad (12)
$$

and $h_k^{(M)}(x) = \left\{ L_k^{(M)}(x) - T \right\}$. We can derive the set of candidate weaker classifiers as

$$\mathcal{H}^{(M)} = \{h_k^{(M)}(x) \mid \forall k\} \tag{13}$$

Recall that the best $h_M(x)$ among all in $\mathcal{H}_{(M-1)}$ for the new strong classifier $H_M(x) = H_{M-1}(x) + h_M(x)$ is given by Eq.(3) among all $h^\dagger \in \mathcal{H}^{(M)}$, for which the optimal weak classifier has been derived as (7). According the theory of gradient based boosting [4, 8, 21], we can choose the optimal weak classifier by finding the $h_M(x)$ that best fits the gradient $-\nabla J(H_{M-1})$ where

$$\nabla J(H_{M-1}) = \left( \frac{\partial J}{\partial x_1}, \cdots, \frac{\partial J}{\partial x_N} \right) \tag{14}$$

In our stagewise approximation formulation, this can be done by first finding the $h_M(x) \in \mathcal{H}^{(M)}$ that best fits $-\nabla J$ in direction and then scaling it so that the two has the same (re-weighted) norm. An alternative selection scheme is simply to choose $k$ so that the error rate (or some risk), computed from the two histograms $p(z_k \mid y = +1, w^{(M-1)})$ and $p(z_k \mid y = -1, w^{(M-1)})$, is minimized.

## 5   Experimental Results

**Face Detection** The face detection problem here is to classifier an image of standard size (eg 20x20 pixels) into either face or nonface (imposter). This is essentially a one-class problem in that everything not a face is a nonface. It is a very hard problem. Learning based methods have been the main approach for solving the problem , eg [13, 16, 9, 12]. Experiments here follow the framework of Viola and Jones [19, 18]. There, AdaBoost is used for learning face detection; it performs two important tasks: feature selection from a large collection features; and constructing classifiers using selected features.

**Data Sets**

A set of 5000 face images are collected from various sources. The faces are cropped and re-scaled to the size of 20x20. Another set of 5000 nonface examples of the same size are collected from images containing no faces. The 5000 examples in each set is divided into a training set of 4000 examples and a test set of 1000 examples. See Fig.3 for a random sample of 10 face and 10 nonface examples.

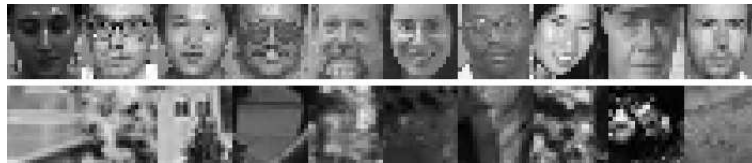

Figure 3: Face (top) and nonface (bottom) examples.

**Scalar Features**

Three basic types of scalar features $z_k$ are derived from each example, as shown in Fig.4, for constructing weak classifiers. These block differences are an extended set of steerable filters used in [10, 20]. There are hundreds of thousands of different $z_k$ for admissible $x, y, dx, dy$ values. Each candidate weak classifier is constructed as the log likelihood ratio (12) computed from the two histograms $p(z_k \mid y, w^{(M-1)})$ of a scalar feature $z_k$ for the face ($y = +1$) and nonface ($y = -1$) examples (*cf.* the last part of the previous section).

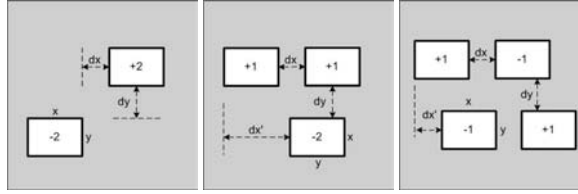

Figure 4: The three types of simple Harr wavelet like features $x_{(j)}$ defined on a sub-window $x$. The rectangles are of size $x \times y$ and are at distances of $(dx, dy)$ apart. Each feature takes a value calculated by the weighted $(\pm 1, 2)$ sum of the pixels in the rectangles.

**Performance Comparison** The same data sets are used for evaluating FloatBoost and AdaBoost. The performance is measured by false alarm error rate given the detection rate fixed at 99.5%. While a cascade of stronger classifiers are needed to achiever very low false alarm [19, 7], here we present the learning curves for the first strong classifier composed of up to one thousand weak classifiers. This is because what we aim to evaluate here is to contrast between FloatBoost and AdaBoost learning algorithms, rather than the system work. Interested reader is referred to [7] for a complete system which achieved a false alarm of $10^{-6}$ with the detection rate of 95%. (A live demo of multi-view face detection system, the first real-time system of the kind in the world, is being submitted to the conference).

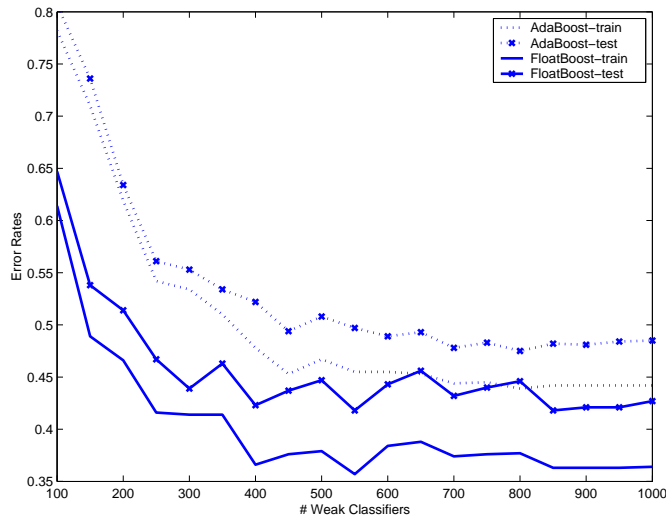

Figure 5: Error Rates of FloatBoost vs AdaBoost for frontal face detection.

The training and testing error curves for FloatBoost and AdaBoost are shown in Fig.5, with the detection rate fixed at 99.5%. The following conclusions can be made from these curves: (1) Given the same number of learned features or weak classifiers, FloatBoost always achieves lower training error and lower test error than AdaBoost. For example, on the test set, by combining 1000 weak classifiers, the false alarm of FloatBoost is 0.427 versus 0.485 of AdaBoost. (2) FloatBoost needs many fewer weak classifiers than AdaBoost in order to achieve the same false alarms. For example, the lowest test error for AdaBoost is 0.481 with 800 weak classifiers, whereas FloatBoost needs only 230 weak classifiers to achieve the same performance. This clearly demonstrates the strength of FloatBoost in

learning to achieve lower error rate.

# 6 Conclusion and Future Work

By incorporating the idea of Floating Search [11] into AdaBoost [3, 15], FloatBoost effectively improves the learning results. It needs fewer weaker classifiers than AdaBoost to achieve a similar error rate, or achieves lower a error rate with the same number of weak classifiers. Such a performance improvement is achieved with the cost of longer training time, about 5 times longer for the experiments reported in this paper.

The Boosting algorithm may need substantial computation for training. Several methods can be used to make the training more efficient with little drop in the training performance. Noticing that only examples with large weigh values are influential, Friedman *et al.* [5] propose to select examples with large weights, *i.e.* those which in the past have been wrongly classified by the learned weak classifiers, for the training weak classifier in t+- he next round. Top examples within a fraction of $1 - \beta$ of the total weight mass are used, where $\beta \in [0.01, 0.1]$.

# References

[1] L. Breiman. "Arcing classifiers". *The Annals of Statistics*, 26(3):801–849, 1998.

[2] P. Buhlmann and B. Yu. "Invited discussion on 'Additive logistic regression: a statistical view of boosting (friedman, hastie and tibshirani)' ". *The Annals of Statistics*, 28(2):377–386, April 2000.

[3] Y. Freund and R. Schapire. "A decision-theoretic generalization of on-line learning and an application to boosting". *Journal of Computer and System Sciences*, 55(1):119–139, Aug 1997.

[4] J. Friedman. "Greedy function approximation: A gradient boosting machine". *The Annals of Statistics*, 29(5), October 2001.

[5] J. Friedman, T. Hastie, and R. Tibshirani. "Additive logistic regression: a statistical view of boosting". *The Annals of Statistics*, 28(2):337–374, April 2000.

[6] M. J. Kearns and U. Vazirani. *An Introduction to Computational Learning Theory*. MIT Press, Cambridge, MA, 1994.

[7] S. Z. Li, L. Zhu, Z. Q. Zhang, A. Blake, H. Zhang, and H. Shum. "Statistical learning of multi-view face detection". In *Proceedings of the European Conference on Computer Vision*, page ???, Copenhagen, Denmark, May 28 - June 2 2002.

[8] L. Mason, J. Baxter, P. Bartlett, and M. Frean. Functional gradient techniques for combining hypotheses. In A. Smola, P. Bartlett, B. Schölkopf, and D. Schuurmans, editors, *Advances in Large Margin Classifiers*, pages 221–247. MIT Press, Cambridge, MA, 1999.

[9] E. Osuna, R. Freund, and F. Girosi. "Training support vector machines: An application to face detection". In *CVPR*, pages 130–136, 1997.

[10] C. P. Papageorgiou, M. Oren, and T. Poggio. "A general framework for object detection". In *Proceedings of IEEE International Conference on Computer Vision*, pages 555–562, Bombay, India, 1998.

[11] P. Pudil, J. Novovicova, and J. Kittler. "Floating search methods in feature selection". *Pattern Recognition Letters*, (11):1119–1125, 1994.

[12] D. Roth, M. Yang, and N. Ahuja. "A snow-based face detector". In *Proceedings of Neural Information Processing Systems*, 2000.

[13] H. A. Rowley, S. Baluja, and T. Kanade. "Neural network-based face detection". *IEEE Transactions on Pattern Analysis and Machine Intelligence*, 20(1):23–28, 1998.

[14] R. Schapire, Y. Freund, P. Bartlett, and W. S. Lee. "Boosting the margin: A new explanation for the effectiveness of voting methods". *The Annals of Statistics*, 26(5):1651–1686, October 1998.

[15] R. E. Schapire and Y. Singer. "Improved boosting algorithms using confidence-rated predictions". In *Proceedings of the Eleventh Annual Conference on Computational Learning Theory*, pages 80–91, 1998.

[16] K.-K. Sung and T. Poggio. "Example-based learning for view-based human face detection". *IEEE Transactions on Pattern Analysis and Machine Intelligence*, 20(1):39–51, 1998.

[17] L. Valiant. "A theory of the learnable". *Communications of ACM*, 27(11):1134–1142, 1984.

[18] P. Viola and M. Jones. "Asymmetric AdaBoost and a detector cascade". In *Proceedings of Neural Information Processing Systems*, Vancouver, Canada, December 2001.

[19] P. Viola and M. Jones. "Rapid object detection using a boosted cascade of simple features". In *Proceedings of IEEE Computer Society Conference on Computer Vision and Pattern Recognition*, Kauai, Hawaii, December 12-14 2001.

[20] P. Viola and M. Jones. "Robust real time object detection". In *IEEE ICCV Workshop on Statistical and Computational Theories of Vision*, Vancouver, Canada, July 13 2001.

[21] R. Zemel and T. Pitassi. "A gradient-based boosting algorithm for regression problems". In *Advances in Neural Information Processing Systems*, volume 13, Cambridge, MA, 2001. MIT Press.
